# Connectionist Approaches to the Use of Markov Models for Speech Recognition

**Hervé Bourlard** [†,‡]
[†] L & H Speechproducts
Koning Albert 1 laan, 64
1780 Wemmel, BELGIUM

**Nelson Morgan** [‡] **& Chuck Wooters** [‡]
[‡] Intl. Comp. Sc. Institute
1947, Center St., Suite 600
Berkeley, CA 94704, USA

## ABSTRACT

Previous work has shown the ability of Multilayer Perceptrons (MLPs) to estimate emission probabilities for Hidden Markov Models (HMMs). The advantages of a speech recognition system incorporating both MLPs and HMMs are the best discrimination and the ability to incorporate multiple sources of evidence (features, temporal context) without restrictive assumptions of distributions or statistical independence. This paper presents results on the speaker-dependent portion of DARPA's English language Resource Management database. Results support the previously reported utility of MLP probability estimation for continuous speech recognition. An additional approach we are pursuing is to use MLPs as nonlinear predictors for autoregressive HMMs. While this is shown to be more compatible with the HMM formalism, it still suffers from several limitations. This approach is generalized to take account of time correlation between successive observations, without any restrictive assumptions about the driving noise.

## 1   INTRODUCTION

We have been working on continuous speech recognition using moderately large vocabularies (1000 words) [1,2]. While some of our research has been in speaker-independent recognition [3], we have primarily used a German speaker-dependent

database called SPICOS [1,2]. In our previously reported work, we developed a hybrid MLP/HMM algorithm in which an MLP is trained to generate the output probabilities of an HMM [1,2]. Given speaker-dependent training, we have been able to recognize 50-60 % of the words in the SPICOS test sentences. While this is not a state-of-the-art level of performance, it was accomplished with single-state phoneme models, no triphone or allophone representations, no function word modeling, etc., and so may be regarded as a "baseline" system. The main point to using such a simple system is simplicity for comparison of the effectiveness of alternate probability estimation techniques. While we are working on extending our technique to more complex systems, the current paper describes the application of the baseline system (with a few changes, such as different VQ features) to the speaker-dependent portion of the English language Resource Management (RM) database (continuous utterances built up from a lexicon of roughly 1000 words) [4]. While this exercise was primarily intended to confirm that the previous result, which showed the utility of MLPs for the estimation of HMM output probabilities, was not restricted to the limited data set of our first experiments, it also shows how to improve further the initial scheme.

However, potential problems remain. In order to improve local discrimination, the MLP is usually provided with contextual inputs [1,2,3] or recurrent links. Unfortunately, in these cases, the dynamic programming recurrences of the Viterbi algorithm are no longer stricly valid when the local probabilities are generated by these contextual MLPs. To solve this problem, we have started considering, as initially proposed in [9] and [10], another approach in which MLP is used as a nonlinear predictor. Along this line, a new approach is suggested and preliminary results are reported.

## 2    METHODS AND RESULTS

As shown by both theoretical [5] and experimental [1] results, MLP output values may be considered to be estimates of a posteriori probabilities. Either these or some other related quantity (such as the output normalized by the prior probability of the corresponding class) may be used in a Viterbi search to determine the best time-warped succession of states to explain the observed speech measurements. This hybrid approach has the potential of exploiting the interpolating capabilities of MLPs while using Dynamic Time Warping (DTW) to capture the dynamics of speech. As described in [2], the practical application of the technique requires cross-validation during training to determine the stopping point, division by the priors at the output to generate likelihoods, optimized word transition penalties, and training sentence alignment via iterations of the Viterbi algorithm.

For the RM data, initial development was done on a single speaker to confirm that the techniques we developed previously [2] were still applicable. Although we experimented slightly with this data, the system we ended up with was substantially unchanged, with the exception of the program modifications required to use different vector quantized (VQ) features. Input features used were based on the front

end for SRI's DECIPHER system [6], including vector quantized mel-cepstrum (12 coefficients), vector-quantized difference of mel-cepstrum, quantized energy, and quantized difference of energy. Both vector quantization codebooks contained 256 prototypes, while energy and delta energy were quantized into 25 levels. A feature vector was calculated for each 10 ms of input speech. Since each feature was represented by a simple binary input vector with only one bit 'on', each 10 ms frame of speech signal was represented by a 562-dimensional binary vector with only 4 bits 'on'. Some experiments were run with no context (i.e., only one frame was input to the network for each classification). To show the advantage of contextual information, other experiments were run with nine frames of input to the network, allowing four frames of contextual information on each side of the current frame being classified. In this case, the input field contained $9 \times 562 = 5058$ units. The size of the output layer was kept fixed at 61 units, corresponding to the 61 phonemes to be recognized. As we found in our SPICOS experiments, a hidden layer was not useful for this problem, probably because of the high dimension of the binary input space and, as a consequence, of the large number of parameters. Of course, it could be argued that a hidden layer should reduce this huge number of parameters, and thus improve generalization. However, networks with no hidden units always outperformed experimental systems with hidden layers, on both the frame and word levels. The ability of the simpler nets to generalize well, despite the sheer number of parameters, was probably due to the cross-validation technique used during the MLP training [7]. However, as shown in [3], hidden layers are useful for the case of continuous input features. In this case, the dimension of the input layer of the MLP is much lower (even with contextual information), so that large hidden layers (e.g., 1000 units) may be useful.

For each speaker, we used 400 sentences for training, 100 for cross-validation, and a final 100 for recognition tests. Starting from an initial segmentation (derived from the average length of the phonemes), a Viterbi algorithm was then iterated with standard emission probabilities (i.e., by counting, no contextual information and assuming independence of the features) to generate a final segmentation which provided us with initial targets for the MLP training. Training of the MLP was done by an error-back propagation algorithm, using an entropy criterion. In each iteration, the complete training set was presented, and the parameters were updated after each training pattern (stochastic gradient). To avoid overtraining of the MLP, improvement on the cross-validation set was checked after each iteration. If the classification rate on the cross-validation set had not improved more than a small threshold, the learning rate of the gradient procedure was reduced by a factor of two. Compared with the results reported in [11], it has been observed recently that it was still possible to improve significantly the recognition performance [11] by starting from a lower initial learning constant and by adapting the segmentation of the training sentences to the MLP. This has been done by using the final segmentation of the standard Viterbi as a new starting point of a Viterbi training embedding now the MLP for estimating the emission probabilities. In this case, each iteration of the Viterbi is followed by a new optimization of the MLP (according to the new

**Table 1:** Word Recognition Performance on RM database

| speaker | ML | MLP(9) | + FWM |
|---------|-----|--------|-------|
| jws04 | 48.2 | 62.3 | |
| bef03 | 39.3 | 56.7 | |
| cmr02 | 59.5 | 70.9 | |
| dtb03 | 49.8 | 61.2 | |
| das12 | 63.8 | 76.5 | 81.8 |
| ers07 | 45.4 | 58.3 | |
| dms04 | 58.0 | 69.1 | |
| tab07 | 60.8 | 70.5 | |
| hxs06 | 60.9 | 76.3 | |
| rkm05 | 37.9 | 53.8 | 60.2 |
| pgh01 | 50.4 | 63.6 | |
| mean | 52.2 | 65.4 | |

segmentation generated by the Viterbi alignment). Recognition performance resulting of this process are reported in the column "MLP(9)" of Table 1. Comparison with results presented in [11] clearly shows the additional improvement (which was also observed at the frame level) that can be gained from such modifications.

## 3    RECOGNITION AND DISCUSSION

For recognition, the output layer of the MLP was evaluated for each frame, and (after division by the prior probability of each phoneme) was used as emission probabilities in a discrete HMM system. In this case, each phoneme $k$ was thus associated with a single conditional density evaluated on the $k$-th output unit of the MLP. In our system, in order to model state duration, each phoneme was modeled by an HMM with a single state $q_k$ repeated $D/2$ times, where $D$ is the prior estimate of the duration of the phoneme as observed on the training set. Only selfloops and sequential transitions were permitted. A Viterbi decoding was then used for recognition of the first thirty sentences of the cross-validation set to optimize word transition probabilities. Note that this same simplified HMM was used for both the Maximum Likelihood (ML) reference system (estimating probabilities directly from relative frequencies) and the MLP system, and that the same input features were used for both.

The first two columns of Table 1 shows the recognition rates (100 % - error rate, where errors include insertions, deletions, and substitutions) for the 100 test sentences of the 11 speakers which were left out in the development, respectively for standard Maximum Likelihood (ML) and MLP with 9 frames of contextual input (MLP(9)). These results (all obtained with no language model, i.e., with a perplexity of 1000 for a 1000 word vocabulary) show the significant improvements that can

be achieved using MLPs for continuous speech recognition (over simpler probability estimators) and that the incorporation of context has a major effect. However, it was also particularly interesting to note that the improvement was already significant with no contextual information at the input [11]. This can be explained by the fact that in standard HMM (denoted ML in Table 1) we must assume the independence of the four features so that we can estimate the joint density by their product, which is not the case with the MLP. This observation was also valid at the frame level [1,11].

However, these results are not the best ones we can expect from such an approach. A way to improve further the performance is to add function word models for small words as it is often done in standard HMMs. This idea was tested by using 28 additional output units (representing 12 word models) to the initial scheme. Results for the best and the worse speaker are reported under the column denoted "+ FWM" in Table 1. In view of the improvements, it can be concluded that many of the tricks valid for standard HMMs are also useful in our approach and can improve significantly the initial results.

## 4   MLP AS AUTOREGRESSIVE MODEL

As shown in the previous Section, it is clear that the proposed HMM/MLP hybrid approach can achieve significant improvements over standard HMMs. However, it has to be observed that these improvements are obtained despite some theoretical weaknesses. Indeed, it can be shown that the Dynamic Programming (DP) recurrences of the Viterbi algorithm (used for training and recognition) are no longer strictly valid when the local probabilities are generated by MLPs with contextual inputs. For a sequence of acoustic vectors $X = \{x_1, \ldots, x_N\}$ and a Markov model $M$, $P(X|M)$ cannot simply be obtained by DP recurrences (which are only valid for first order Markov models) using the contextual MLP outputs (divided by the priors). Thus, neither feedback or contextual input to the MLP (followed by the Bayes' rule to estimate $P(X|M)$) are stricly correct to use for the Viterbi algorithm, since both violate the restriction to instantaneous features on the left side of the conditional in local probabilities (in our case, the system is even not causal any more). This problem does not appear in standard HMMs were contextual information is usually provided via dynamic features such as the first and second derivatives (which are, in theory, estimates of instantaneous features) of the time-varying acoustic vectors.

In [9] and [10], another approach, related to autoregressive (AR) HMMs [8], is proposed in which the MLP is used as a nonlinear predictor. The basic idea is to assume that the observed vectors associated with each HMM state are drawn from a particular AR process described by an AR function that can be linear [8] or nonlinear and associated with the transfer function of an MLP. If $x_n$ is the acoustic vector at time $n$ and if $X_{n-p}^{n-1} = \{x_{n-p}, \ldots, x_{n-1}\}$ denotes the input of the MLP (which attempts to predict $x_n$, the desired output of the MLP associated with $X_{n-p}^{n-1}$), it can be shown [8,9,12] that, if the prediction error is assumed to be Gaussian with zero mean and unity variance, minimization of the prediction

error is equivalent to estimation of $p(x_n|q_k^n, X_{n-p}^{n-1})$ (where $q_k^n$ is the HMM state associated with $x_n$), which can be expressed as a Gaussian (with unity variance) where the exponent is the prediction error. Consequently, the prediction errors can be used as local distances in DP and are fully compatible with the recurrences of the Viterbi algorithm. However, although the MLP/HMM interface problem seems to be solved, we are now limited to Gaussian AR processes. Furthermore, each state must be associated with its own MLP [10]. An alternative approach, as proposed in [9], is to have a single MLP with additional "control" inputs coding the state being considered. However, in both cases, the discriminant character of the MLP is lost since it is only used as a nonlinear predictor. On preliminary experiments on SPICOS we were unable to get significant results from these approaches compared with the method presented in the previous Section [1,2].

However, it is possible to generalize the former approach and to avoid the Gaussian hypothesis. It is indeed easy to prove (by using Bayes' rule with an additional conditional $X_{n-p}^{n-1}$ everywhere) that:

$$p(x_n|q_k^n, X_{n-p}^{n-1}) = \frac{p(q_k^n|X_{n-p}^n)\, p(x_n|X_{n-p}^{n-1})}{p(q_k^n|X_{n-p}^{n-1})} . \qquad (1)$$

As $p(x_n|X_{n-p}^{n-1})$ in (1) is independent of the classes $q_k$ it can overlooked in the DP recurrences. In this case, without any assumption about mean and covariance of the driving noise, $p(x_n|q_k^n, X_{n-p}^{n-1})$ can be expressed as the ratio of the output values of two "standard" MLPs (as used in the previous Section and in [1,2]), respectively with $X_{n-p}^{n-1}$ and $X_{n-p}^n$ as input. In preliminary experiments, this approach lead to better results then the former AR models without however bearing comparison with the method used in the previous Section and in [1,2]. For example, on SPICOS and after tuning, we got 46 % recognition rate instead of 65 % with our best method [2].

## 5  CONCLUSION

Despite some theoretical nonidealities, the HMM/MLP hybrid approach can achieve significant improvement over comparable standard HMMs. This was observed using a simplified HMM system with single-state monophone models, and no langauge model. However, the reported results also show that many of the tricks used to improve standard HMMs are also valid for our hybrid approach, which leaves the way open to all sort of further developments. Now that we have confirmed the principle, we are beginning to develop a complete system, which will incorporate context-dependent sound units. In this framework, we are studying the possibility of modeling multi-states HMMs and triphones. On the other hand, in spite of preliminary disappointing performance (which seems to corroborate previous experiments done by others [13,14] with AR processes for speech recognition), MLPs as AR models are still worth considering further given their attractive theoretical basis and better interface with the HMM formalism.

## References

[1] Bourlard, H., Morgan, N., & Wellekens, C.J., "Statistical Inference in Multilayer Perceptrons and Hidden Markov Models with Applications in Continuous Speech Recognition", *Neurocomputing*, Ed. F. Fogelman & J. Hérault, NATO ASI Series, vol. F68, pp. 217-226, 1990.

[2] Morgan, N., & Bourlard, H., "Continuous Speech Recognition using Multilayer Perceptrons with Hidden Markov Models", *IEEE Proc. of the 1990 Intl. Conf. on ASSP*, pp. 413-416, Albuquerque, NM, April 1990.

[3] Morgan, N., Hermansky, H., Bourlard, H., Kohn, P., Wooters, C., & Kohn, P., "Continuous Speech Recognition Using PLP Analysis with Multilayer Perceptrons" accepted for *IEEE Proc. of the 1991 Intl. Conf. on ASSP*, Toronto, 1991.

[4] Price, P., Fisher, W., Bernstein, J., & Pallet, D., "The DARPA 1000-Word Resource Management Database for Continuous Speech Recognition", *Proc. IEEE Intl. Conf. on ASSP*, pp. 651-654, New-York, 1988.

[5] Bourlard, H., & Wellekens, C.J., "Links between Markov Models and Multilayer Perceptrons", *IEEE Trans. on Pattern Analysis and Machine Intelligence*, Vol. 12, No. 12, pp. 1167-1178, December 1990.

[6] Murveit, H., & Weintraub, M., "1000-Word Speaker-Independent Continuous Speech Recognition Using Hidden Markov Models", *Proc. IEEE Intl. Conf. on ASSP*, pp. 115-118, New-York, 1988.

[7] Morgan, N., & Bourlard, H., "Generalization and Parameter Estimation in Feedforward Nets: Some Experiments", *Advances in Neural Information Processing Systems 2*, Ed. D.S Touretzky, San Mateo, CA: Morgan-Kaufmann, pp. 630-637, 1990.

[8] Juang, B.H. & Rabiner, L.R., "Mixture Autoregressive Hidden Markov Models for Speech Signals", *IEEE Trans. on ASSP*, vol. 33, no. 6, pp. 1404-1412, 1985.

[9] Levin, E., "Speech Recognition Using Hidden Control Neural Network Architecture", *Proc. of IEEE Intl. Conf. on ASSP*, Albuquerque, New Mexico, 1990.

[10] Tebelskis, J., & Waibel A., "Large Vocabulary Recognition Using Linked Predictive Neural Networks", *Proc. of IEEE Intl. Conf. on ASSP*, Albuquerque, New Mexico, 1990.

[11] Morgan, N., Wooters, C., Bourlard, H., & Cohen, M., "Continuous Speech Recognition on the Resource Management Database Using Connectionist Probability Estimation", *Proc. of Intl. Conf. on Spoken Language Processing*, Kobe, Japan, 1990.

[12] Bourlard, H., "How Connectionist Models Could Improve Markov Models for Speech Recognition", *Advanced Neural Computers*, Ed. R. Eckmiller, North-Holland, pp. 247-254, 1990.

[13] de La Noue, P., Levinson, S., & Sondhi M., "Incorporating the Time Correlation Between Successive Observations in an Acoustic-Phonetic Hidden Markov Model for Continuous Speech Recognition", AT&T Technical Memorandum No. 11226, 1989.

[14] Wellekens, C.J., "Explicit Time Correlation in Hidden Markov Models", *Proc. of the IEEE Intl. Conf. on ASSP*, Dallas, Texas, 1987.
